# Playing is believing:
# The role of beliefs in multi-agent learning

**Yu-Han Chang**
Artificial Intelligence Laboratory
Massachusetts Institute of Technology
Cambridge, Massachusetts 02139
*ychang@ai.mit.edu*

**Leslie Pack Kaelbling**
Artificial Intelligence Laboratory
Massachusetts Institute of Technology
Cambridge, Massachusetts 02139
*lpk@ai.mit.edu*

## Abstract

We propose a new classification for multi-agent learning algorithms, with each *league* of players characterized by both their possible strategies and possible beliefs. Using this classification, we review the optimality of existing algorithms, including the case of interleague play. We propose an incremental improvement to the existing algorithms that seems to achieve average payoffs that are at least the Nash equilibrium payoffs in the long-run against *fair* opponents.

## 1 Introduction

The topic of learning in multi-agent environments has received increasing attention over the past several years. Game theorists have begun to examine learning models in their study of repeated games, and reinforcement learning researchers have begun to extend their single-agent learning models to the multiple-agent case. As traditional models and methods from these two fields are adapted to tackle the problem of multi-agent learning, the central issue of optimality is worth revisiting. What do we expect a successful learner to do?

**Matrix games and Nash equilibrium.** From the game theory perspective, the repeated game is a generalization of the traditional one-shot game, or *matrix game*. The matrix game is defined as a reward matrix $R_i$ for each player, $R_i : A_1 \times A_2 \to \mathbb{R}$, where $A_i$ is the set of actions available to player $i$. Purely competitive games are called *zero-sum games* and must satisfy $R_1 = -R_2$. Each player simultaneously chooses to play a particular action $a_i \in A_i$, or a mixed policy $\mu_i = PD(A_i)$, which is a probability distribution over the possible actions, and receives reward based on the joint action taken. Some common examples of single-shot matrix games are shown in Figure 1. The traditional assumption is that each player has no prior knowledge about the other player. As is standard in the game theory literature, it is thus reasonable to assume that the opponent is fully rational and chooses actions that are in its best interest. In return, we must play a *best response* to the opponent's choice of action. A *best response* function for player $i$, $BR_i(\mu_{-i})$, is defined to be the set of all optimal policies for player $i$, given that the other players are playing the joint policy $\mu_{-i}$: $BR_i(\mu_{-i}) = \{\mu_i^* \in M_i | R_i(\mu_i^*, \mu_{-i}) \geq R_i(\mu_i, \mu_{-i}) \forall \mu_i \in M_i\}$, where $M_i$ is the set of all possible policies for agent $i$.

If all players are playing best responses to the other players' strategies, $\mu_i \in BR_i(\mu_{-i}) \forall i$,

$$R_1 = \begin{bmatrix} -1 & 1 \\ 1 & -1 \end{bmatrix} \quad R_1 = \begin{bmatrix} 0 & -1 & 1 \\ 1 & 0 & -1 \\ -1 & 1 & 0 \end{bmatrix} \quad R_1 = \begin{bmatrix} 0 & 3 \\ 1 & 2 \end{bmatrix} \quad R_1 = \begin{bmatrix} 2 & 0 \\ 3 & 1 \end{bmatrix}$$

$$R_2 = -R_1 \qquad\qquad R_2 = -R_1 \qquad\qquad R_2 = \begin{bmatrix} 0 & 1 \\ 3 & 2 \end{bmatrix} \quad R_2 = \begin{bmatrix} 2 & 3 \\ 0 & 1 \end{bmatrix}$$

(a) Matching pennies     (b) Rock-Paper-Scissors     (c) Hawk-Dove     (d) Prisoner's Dilemna

Figure 1: Some common examples of single-shot matrix games.

then the game is said to be in *Nash equilibrium*. Once all players are playing by a Nash equilibrium, no single player has an incentive to unilaterally deviate from his equilibrium policy. Any game can be solved for its Nash equilibria using quadratic programming, and a player can choose an optimal strategy in this fashion, given prior knowledge of the game structure. The only problem arises when there are multiple Nash equilibria. If the players do not manage to coordinate on one equilibrium joint policy, then they may all end up worse off. The Hawk-Dove game shown in Figure 1(c) is a good example of this problem. The two Nash equilibria occur at (1,2) and (2,1), but if the players do not coordinate, they may end up playing a joint action (1,1) and receive 0 reward.

**Stochastic games and reinforcement learning.** Despite these problems, there is general agreement that Nash equilibrium is an appropriate solution concept for one-shot games. In contrast, for repeated games there are a range of different perspectives. Repeated games generalize one-shot games by assuming that the players repeat the matrix game over many time periods. Researchers in reinforcement learning view repeated games as a special case of stochastic, or Markov, games. Researchers in game theory, on the other hand, view repeated games as an extension of their theory of one-shot matrix games. The resulting frameworks are similar, but with a key difference in their treatment of game history. Reinforcement learning researchers focus their attention on choosing a single stationary policy $\mu$ that will maximize the learner's expected rewards in all future time periods given that we are in time $t$, $\max_\mu E_\mu \left[ \sum_{\tau=t}^{T} \gamma^{\tau-t} R^\tau(\mu) \right]$, where $T$ may be finite or infinite, and $\mu = PD(A)$. In the infinite time-horizon case, we often include the discount factor $0 < \gamma < 1$.

Littman [1] analyzes this framework for zero-sum games, proving convergence to the Nash equilibrium for his minimax-Q algorithm playing against another minimax-Q agent. Claus and Boutilier [2] examine cooperative games where $R_1 = R_2$, and Hu and Wellman [3] focus on general-sum games. These algorithms share the common goal of finding and playing a Nash equilibrium. Littman [4] and Hall and Greenwald [5] further extend this approach to consider variants of Nash equilibrium for which convergence can be guaranteed. Bowling and Veloso [6] and Nagayuki et al. [7] propose to relax the mutual optimality requirement of Nash equilibrium by considering *rational* agents, which always learn to play a stationary best-response to their opponent's strategy, even if the opponent is not playing an equilibrium strategy. The motivation is that it allows our agents to act rationally even if the opponent is not acting rationally because of physical or computational limitations. Fictitious play [8] is a similar algorithm from game theory.

**Game theoretic perspective of repeated games.** As alluded to in the previous section, game theorists often take a more general view of optimality in repeated games. The key difference is the treatment of the history of actions taken in the game. Recall that in the

Table 1: Summary of multi-agent learning algorithms under our new classification.

| | $\mathcal{B}_0$ | $\mathcal{B}_1$ | $\mathcal{B}_\infty$ |
|---|---|---|---|
| $\mathcal{H}_0$ | minimax-Q, Nash-Q | | Bully |
| $\mathcal{H}_1$ | | | Godfather |
| $\mathcal{H}_\infty$ | $Q$-learning ($Q_0$), (WoLF-)PHC, fictitious play | $Q_1$ | multiplicative-weight* |

* assumes public knowledge of the opponent's policy at each period

stochastic game model, we took $\mu_i = PD(A_i)$. Here we redefine $\mu_i : H \rightarrow PD(A_i)$, where $H = \bigcup_t H^t$ and $H^t$ is the set of all possible histories of length $t$. Histories are observations of joint actions, $h^t = (a_i, a_{-i}, h^{t-1})$. Player $i$'s strategy at time $t$ is then expressed as $\mu_i(h^{t-1})$. In essence, we are endowing our agent with memory. Moreover, the agent ought to be able to form beliefs about the opponent's strategy, and these beliefs ought to converge to the opponent's actual strategy given sufficient learning time. Let $\beta_i : H \rightarrow PD(A_{-i})$ be player $i$'s belief about the opponent's strategy. Then a learning path is defined to be a sequence of histories, beliefs, and personal strategies. Now we can define the Nash equilibrium of a repeated game in terms of our personal strategy and our beliefs about the opponent. If our prediction about the opponent's strategy is accurate, then we can choose an appropriate best-response strategy. If this holds for all players in the game, then we are guaranteed to be in Nash equilibrium.

**Proposition 1.1.** A learning path $\{(h^t, \mu_i(h^{t-1}), \beta_i(h^{t-1}))|t = 1, 2, \ldots\}$ converges to a Nash equilibrium iff the following two conditions hold:

- Optimization: $\forall t, \mu_i(h^{t-1}) \in BR_i(\beta_i(h^{t-1}))$. We always play a best-response to our prediction of the opponent's strategy.

- Prediction: $\lim_{t\rightarrow\infty} |\beta_i(h^{t-1}) - \mu_{-i}(h^{t-1})| = 0$. Over time, our belief about the opponent's strategy converges to the opponent's actual strategy.

However, Nachbar and Zame [9] shows that this requirement of simultaneous prediction and optimization is impossible to achieve, given certain assumptions about our possible strategies and possible beliefs. We can never design an agent that will learn to both predict the opponent's future strategy and optimize over those beliefs at the same time. Despite this fact, if we assume some extra knowledge about the opponent, we can design an algorithm that approximates the best-response stationary policy over time against *any* opponent. In the game theory literature, this concept is often called *universal consistency*. Fudenburg and Levine [8] and Freund and Schapire [10] independently show that a multiplicative-weight algorithm exhibits universal consistency from the game theory and machine learning perspectives. This give us a strong result, but requires the strong assumption that we know the opponent's policy at each time period. This is typically not the case.

## 2   A new classification and a new algorithm

We propose a general classification that categorizes algorithms by the cross-product of their possible strategies and their possible beliefs about the opponent's strategy, $\mathcal{H} \times \mathcal{B}$. An agent's possible strategies can be classified based upon the amount of history it has in memory, from $\mathcal{H}_0$ to $\mathcal{H}_\infty$. Given more memory, the agent can formulate more complex policies, since policies are maps from histories to action distributions. $\mathcal{H}_0$ agents are memoryless and can only play stationary policies. Agents that can recall the actions from the previous

time period are classified as $\mathcal{H}_1$ and can execute reactive policies. At the other extreme, $\mathcal{H}_\infty$ agents have unbounded memory and can formulate ever more complex strategies as the game is played over time. An agent's belief classification mirrors the strategy classification in the obvious way. Agents that believe their opponent is memoryless are classified as $\mathcal{B}_0$ players, $\mathcal{B}_t$ players believe that the opponent bases its strategy on the previous $t$-periods of play, and so forth. Although not explicitly stated, most existing algorithms make assumptions and thus hold beliefs about the types of possible opponents in the world.

We can think of each $\mathcal{H}_s \times \mathcal{B}_t$ as a different *league* of players, with players in each league roughly equal to one another in terms of their capabilities. Clearly some leagues contain less capable players than others. We can thus define a *fair* opponent as an opponent from an equal or lesser league. The idea is that new learning algorithms should ideally be designed to beat any fair opponent.

**The key role of beliefs.**   Within each league, we assume that players are fully rational in the sense that they can fully use their available histories to construct their future policy. However, an important observation is that the definition of full rationality depends on their beliefs about the opponent. If we believe that our opponent is a memoryless player, then even if we are an $\mathcal{H}_\infty$ player, our fully rational strategy is to simply model the opponent's stationary strategy and play our stationary best response. Thus, our belief capacity and our history capacity are inter-related. Without a rich set of possible beliefs about our opponent, we cannot make good use of our available history. Similarly, and perhaps more obviously, without a rich set of historical observations, we cannot hope to model complex opponents.

**Discussion of current algorithms.**   Many of the existing algorithms fall within the $\mathcal{H}_\infty \times \mathcal{B}_0$ league. As discussed in the previous section, the problem with these players is that even though they have full access to the history, their fully rational strategy is stationary due to their limited belief set. A general example of a $\mathcal{H}_\infty \times \mathcal{B}_0$ player is the policy hill climber (PHC). It maintains a policy and updates the policy based upon its history in an attempt to maximize its rewards. Originally PHC was created for stochastic games, and thus each policy also depends on the current state $s$. In our repeated games, there is only one state.

For agent $i$, Policy Hill Climbing (PHC) proceeds as follows:

1. Let $\alpha$ and $\delta$ be the learning rates. Initialize

$$Q(s, a) \leftarrow 0, \mu_i(s, a) \leftarrow \frac{1}{|A_i|} \forall s \in S, a \in A_i.$$

2. Repeat,

a. From state $s$, select action $a$ according to the mixed policy $\mu_i(s)$ with some exploration.

b. Observing reward $r$ and next state $s'$, update

$$Q(s, a) \leftarrow (1 - \alpha)Q(s, a) + \alpha(r + \gamma \max_{a'} Q(s', a')).$$

c. Update $\mu(s, a)$ and constrain it to a legal probability distribution:

$$\mu_i(s, a) \leftarrow \mu_i(s, a) + \begin{cases} \delta & \text{if } a = \text{argmax}_{a'} Q(s, a') \\ \frac{-\delta}{|A_i|-1} & \text{otherwise} \end{cases}.$$

The basic idea of PHC is that the $Q$-values help us to define a gradient upon which we execute hill-climbing. Bowling and Veloso's WoLF-PHC [6] modifies PHC by adjusting $\delta$ depending on whether the agent is "winning" or "losing." True to their league, PHC players play well against stationary opponents.

At the opposite end of the spectrum, Littman and Stone [11] propose algorithms in $\mathcal{H}_0 \times \mathcal{B}_\infty$ and $\mathcal{H}_1 \times \mathcal{B}_\infty$ that are *leader* strategies in the sense that they choose a fixed strategy and hope that their opponent will "follow" by learning a best response to that fixed strategy. Their "Bully" algorithm chooses a fixed memoryless stationary policy, while "Godfather" has memory of the last time period. Opponents included normal $Q$-learning and $Q_1$ players, which are similar to $Q$-learners except that they explicitly learn using one period of memory because they believe that their opponent is also using memory to learn. The interesting result is that "Godfather" is able to achieve non-stationary equilibria against $Q_1$ in the repeated prisoner's dilemma game, with rewards for both players that are higher than the stationary Nash equilibrium rewards. This demonstrates the power of having belief models. However, because these algorithms do not have access to more than one period of history, they cannot begin to attempt to construct statistical models the opponent. "Godfather" works well because it has a built-in best response to $Q_1$ learners rather than attempting to learn a best response from experience.

Finally, Hu and Wellman's Nash-Q and Littman's minimax-Q are classified as $\mathcal{H}_0 \times \mathcal{B}_0$ players, because even though they attempt to learn the Nash equilibrium through experience, their play is fixed once this equilibrium has been learned. Furthermore, they assume that the opponent also plays a fixed stationary Nash equilibrium, which they hope is the other half of their own equilibrium strategy. These algorithms are summarized in Table 1.

**A new class of players.**    As discussed above, most existing algorithms do not form beliefs about the opponent beyond $\mathcal{B}_0$. None of these approaches is able to capture the essence of game-playing, which is a world of threats, deceits, and generally out-witting the opponent. We wish to open the door to such possibilities by designing learners that can model the opponent and use that information to achieve better rewards. Ideally we would like to design an algorithm in $\mathcal{H}_\infty \times \mathcal{B}_\infty$ that is able to win or come to an equilibrium against any fair opponent. Since this is impossible [9], we start by proposing an algorithm in the league $\mathcal{H}_\infty \times \mathcal{B}_\infty$ that plays well against a restricted class of opponents. Since many of the current algorithms are best-response players, we choose an opponent class such as PHC, which is a good example of a best-response player in $\mathcal{H}_\infty \times \mathcal{B}_0$. We will demonstrate that our algorithm indeed beats its PHC opponents and in fact does well against most of the existing fair opponents.

**A new algorithm: PHC-Exploiter.**    Our algorithm is different from most previous work in that we are explicitly modeling the opponent's learning algorithm and not simply his current policy. In particular, we would like to model players from $\mathcal{H}_\infty \times \mathcal{B}_0$. Since we are in $\mathcal{H}_\infty \times \mathcal{B}_\infty$, it is rational for us to construct such models because we believe that the opponent is learning and adapting to us over time using its history. The idea is that we will "fool" our opponent into thinking that we are stupid by playing a decoy policy for a number of time periods and then switch to a different policy that takes advantage of their best response to our decoy policy. From a learning perspective, the idea is that we adapt much faster than the opponent; in fact, when we switch away from our decoy policy, our adjustment to the new policy is immediate. In contrast, the $\mathcal{H}_\infty \times \mathcal{B}_0$ opponent adjusts its policy by small increments and is furthermore unable to model our changing behavior. We can repeat this "bluff and bash" cycle ad infinitum, thereby achieving infinite total rewards as $t \to \infty$. The opponent never catches on to us because it believes that we only play stationary policies.

A good example of a $\mathcal{H}_\infty \times \mathcal{B}_0$ player is PHC. Bowling and Veloso showed that in self-play, a restricted version of WoLF-PHC always reaches a stationary Nash equilibrium in two-player two-action games, and that the general WoLF-PHC seems to do the same in experimental trials. Thus, in the long run, a WoLF-PHC player achieves its stationary Nash equilibrium payoff against any other PHC player. We wish to do better than that by exploiting our knowledge of the PHC opponent's learning strategy. We can construct

a *PHC-Exploiter* algorithm for agent $i$ that proceeds like PHC in steps 1-2b, and then continues as follows:

c. Observing action $a^t_{-i}$ at time $t$, update our history $h$ and calculate an estimate of the opponent's policy:

$$\hat{\mu}^t_{-i}(s, a) = \frac{\sum_{\tau=t-w}^{t} \#(h[\tau] = a)}{w} \; \forall a,$$

where $w$ is the window of estimation and $\#(h[\tau] = a) = 1$ if the opponent's action at time $\tau$ is equal to $a$, and 0 otherwise. We estimate $\hat{\mu}^{t-w}_{-i}(s)$ similarly.

d. Update $\delta$ by estimating the learning rate of the PHC opponent:

$$\delta \leftarrow \frac{\left| \hat{\mu}^t_{-i}(s) - \hat{\mu}^{t-w}_{-i}(s) \right|}{w}.$$

e. Update $\mu_i(s, a)$. If we are winning, i.e. $\sum_{a'} \mu_i(s, a')Q(s, a') > R_i(\hat{\mu}^*_i(s), \hat{\mu}_{-i}(s))$, then update

$$\mu_i(s, a) \leftarrow \begin{cases} 1 & \text{if } a = \text{argmax}_{a'} \, Q(s, a') \\ 0 & \text{otherwise} \end{cases},$$

otherwise we are losing, then update

$$\mu_i(s, a) \leftarrow \mu_i(s, a) + \begin{cases} \delta & \text{if } a = \text{argmax}_{a'} \, Q(s, a') \\ \frac{-\delta}{|A_i|-1} & \text{otherwise} \end{cases}.$$

Note that we derive both the opponent's learning rate $\delta$ and the opponent's policy $\hat{\mu}_{-i}(s)$ from estimates using the observable history of actions. If we assume the game matrix is public information, then we can solve for the equilibrium strategy $\hat{\mu}^*_i(s)$, otherwise we can run WoLF-PHC for some finite number of time periods to obtain an estimate this equilibrium strategy. The main idea of this algorithm is that we take full advantage of all time periods in which we are winning, that is, when $\sum_{a'} \mu_i(s, a')Q(s, a') > R_i(\hat{\mu}^*_i(s), \hat{\mu}_{-i}(s))$.

**Analysis.**   The PHC-Exploiter algorithm is based upon PHC and thus exhibits the same behavior as PHC in games with a single pure Nash equilibrium. Both agents generally converge to the single pure equilibrium point. The interesting case arises in competitive games where the only equilibria require mixed strategies, as discussed by Singh et al [12] and Bowling and Veloso [6]. Matching pennies, shown in Figure 1(a), is one such game. PHC-Exploiter is able to use its model of the opponent's learning algorithm to choose better actions.

In the full knowledge case where we know our opponent's policy $\mu_2$ and learning rate $\delta_2$ at every time period, we can prove that a PHC-Exploiter learning algorithm will guarantee us unbounded reward in the long run playing games such as matching pennies.

**Proposition 2.1.**   In the zero-sum game of matching pennies, where the only Nash equilibrium requires the use of mixed strategies, PHC-Exploiter is able to achieve unbounded rewards as $t \to \infty$ against any PHC opponent given that play follows the cycle $C$ defined by the arrowed segments shown in Figure 2.

Play proceeds along $C_w$, $C_l$, then jumps from $(0.5, 0)$ to $(1,0)$, follows the line segments to $(0.5, 1)$, then jumps back to $(0, 1)$. Given a point $(x, y) = (\mu_1(H), \mu_2(H))$ on the graph in Figure 2, where $\mu_i(H)$ is the probability by which player $i$ plays Heads, we know that our expected reward is

$$R_1(x, y) = -1 \times [(x)(y) + (1 - x)(1 - y)] + 1 \times [(1 - x)(y) + (x)(1 - y)].$$

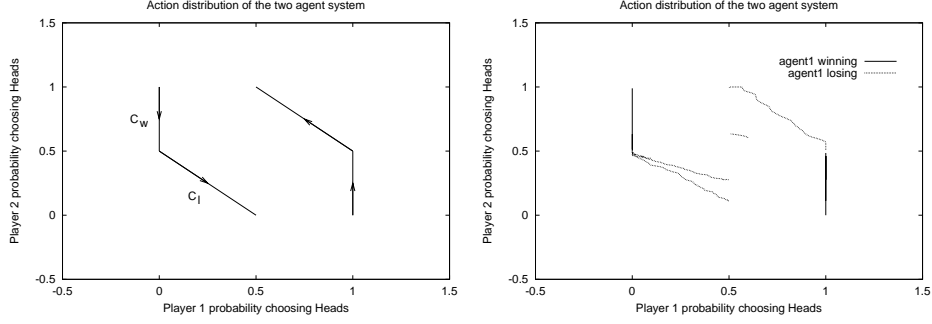

Figure 2: Theoretical (left), Empirical (right). The cyclic play is evident in our empirical results, where we play a PHC-Exploiter player 1 against a PHC player 2.

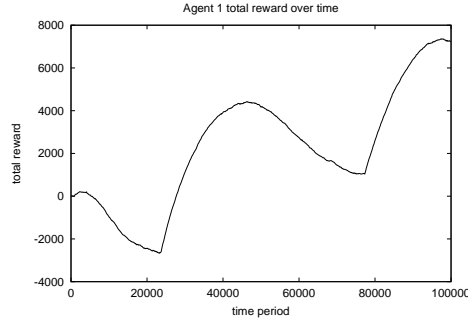

Figure 3: Total rewards for agent 1 increase as we gain reward through each cycle.

We wish to show that

$$\int_C R_1(x,y)dt = 2 \times \left( \int_{C_w} R_1(x,y)dt + \int_{C_l} R_1(x,y)dt \right) > 0 \quad .$$

We consider each part separately. In the losing section, we let $g(t) = x = t$ and $h(t) = y = 1/2 - t$, where $0 \le t \le 1/2$. Then

$$\int_{C_l} R_1(x,y)dt = \int_0^{1/2} R_1(g(t),h(t))dt = -\frac{1}{12} \quad .$$

Similarly, we can show that we receive 1/4 reward over $C_w$. Thus, $\int_C R_1(x,y)dt = 1/3 > 0$, and we have shown that we receive a payoff greater than the Nash equilibrium payoff of zero over every cycle. It is easy to see that play will indeed follow the cycle $C$ to a good approximation, depending on the size of $\delta_2$. In the next section, we demonstrate that we can estimate $\mu_2$ and $\delta_2$ sufficiently well from past observations, thus eliminating the full knowledge requirements that were used to ensure the cyclic nature of play above.

**Experimental results.** We used the PHC-Exploiter algorithm described above to play against several PHC variants in different iterated matrix games, including matching pennies, prisoner's dilemma, and rock-paper-scissors. Here we give the results for the matching pennies game analyzed above, playing against WoLF-PHC. We used a window of $w = 5000$ time periods to estimate the opponent's current policy $\mu_2$ and the opponent's

learning rate $\delta_2$. As shown in Figure 2, the play exhibits the cyclic nature that we predicted. The two solid vertical lines indicate periods in which our PHC-Exploiter player is winning, and the dashed, roughly diagonal, lines indicate periods in which it is losing.

In the analysis given in the previous section, we derived an upper bound for our total rewards over time, which was 1/6 for each time step. Since we have to estimate various parameters in our experimental run, we do not achieve this level of reward. We gain an average of 0.08 total reward for each time period. Figure 3 plots the total reward for our PHC-Exploiter agent over time. The periods of winning and losing are very clear from this graph. Further experiments tested the effectiveness of PHC-Exploiter against other fair opponents, including itself. Against all the existing fair opponents shown in Table 1, it achieved at least its average equilibrium payoff in the long-run. Not surprisingly, it also posted this score when it played against a multiplicative-weight learner.

**Conclusion and future work.** In this paper, we have presented a new classification for multi-agent learning algorithms and suggested an algorithm that seems to dominate existing algorithms from the fair opponent leagues when playing certain games. Ideally, we would like to create an algorithm in the league $\mathcal{H}_\infty \times \mathcal{B}_\infty$ that provably dominates larger classes of fair opponents in any game. Moreover, all of the discussion contained within this paper dealt with the case of iterated matrix games. We would like to extend our framework to more general stochastic games with multiple states and multiple players. Finally, it would be interesting to find practical applications of these multi-agent learning algorithms.

**Acknowledgements.** This work was supported in part by a Graduate Research Fellowship from the National Science Foundation.

# References

[1] Michael L. Littman. Markov games as a framework for multi-agent reinforcement learning. In *Proceedings of the 11th International Conference on Machine Learning (ICML-94)*, 1994.

[2] Caroline Claus and Craig Boutilier. The dynamics of reinforcement learning in cooperative multiaent systems. In *Proceedings of the 15th Natl. Conf. on Artificial Intelligence*, 1998.

[3] Junling Hu and Michael P. Wellman. Multiagent reinforcement learning: Theoretical framework and an algorithm. In *Proceedings of the 15th Int. Conf. on Machine Learning (ICML-98)*, 1998.

[4] Michael L. Littman. Friend-or-foe q-learning in general-sum games. In *Proceedings of the 18th Int. Conf. on Machine Learning (ICML-01)*, 2001.

[5] Keith Hall and Amy Greenwald. Correlated q-learning. In *DIMACS Workshop on Computational Issues in Game Theory and Mechanism Design*, 2001.

[6] Michael Bowling and Manuela Veloso. Multiagent learning using a variable learning rate. Under submission.

[7] Yasuo Nagayuki, Shin Ishii, and Kenji Doya. Multi-agent reinforcement learning: An approach based on the other agent's internal model. In *Proceedings of the International Conference on Multi-Agent Systems (ICMAS-00)*, 2000.

[8] Drew Fudenburg and David K. Levine. Consistency and cautious fictitious play. *Journal of Economic Dynamics and Control*, 19:1065–1089, 1995.

[9] J.H. Nachbar and W.R. Zame. Non-computable strategies and discounted repeated games. *Economic Theory*, 1996.

[10] Yoav Freund and Robert E. Schapire. Adaptive game playing using multiplicative weights. *Games and Economic Behavior*, 29:79–103, 1999.

[11] Michael Littman and Peter Stone. Leading best-response stratgies in repeated games. In *17th Int. Joint Conf. on Artificial Intelligence (IJCAI-2001) workshop on Economic Agents, Models, and Mechanisms*, 2001.

[12] S. Singh, M. Kearns, and Y. Mansour. Nash convergence of gradient dynamics in general-sum games. In *Proceedings of the 16th Conference on Uncertainty in Artificial Intelligence*, 2000.
